# Wiring optimization in the brain

**Dmitri B. Chklovskii**
Sloan Center for
Theoretical Neurobiology
The Salk Institute
La Jolla, CA 92037
*mitya@salk.edu*

**Charles F. Stevens**
Howard Hughes Medical Institute
and Molecular Neurobiology Lab
The Salk Institute
La Jolla, CA 92037
*stevens@salk.edu*

## Abstract

The complexity of cortical circuits may be characterized by the number of synapses per neuron. We study the dependence of complexity on the fraction of the cortical volume that is made up of "wire" (that is, of axons and dendrites), and find that complexity is maximized when wire takes up about 60% of the cortical volume. This prediction is in good agreement with experimental observations. A consequence of our arguments is that any rearrangement of neurons that takes more wire would sacrifice computational power.

Wiring a brain presents formidable problems because of the extremely large number of connections: a microliter of cortex contains approximately $10^5$ neurons, $10^9$ synapses, and 4 km of axons, with 60% of the cortical volume being taken up with "wire", half of this by axons and the other half by dendrites.[1] Each cortical neighborhood must have exactly the right balance of components; if too many cell bodies were present in a particular mm cube, for example, insufficient space would remain for the axons, dendrites and synapses. Here we ask "What fraction of the cortical volume should be wires (axons + dendrites)?" We argue that physiological properties of axons and dendrites dictate an optimal wire fraction of 0.6, just what is actually observed.

To calculate the optimal wire fraction, we start with a real cortical region containing a fixed number of neurons, a mm cube, for example, and imagine perturbing it by adding or subtracting synapses and the axons and dendrites needed to support them. The rules for perturbing the cortical cube require that the existing circuit connections and function remain intact (except for what may have been removed in the perturbation), that no holes are created, and that all added (or subtracted) synapses are typical of those present; as wire volume is added, the volume of the cube of course increases. The ratio of the number of synapses per neuron in the perturbed cortex to that in the real cortex is denoted by $\theta$, a parameter we call the *relative complexity*. We require that the volume of non-wire components (cell bodies, blood vessels, glia, etc) is unchanged by our perturbation and use $\phi$ to denote the volume fraction of the perturbed cortical region that is made up of wires (axons + dendrites; $\phi$ can vary between zero and one), with the fraction for the real brain being $\phi_0$. The relation between relative complexity $\theta$ and wire volume fraction $\phi$ is given by the equation (derived in Methods)

$$\theta = \frac{1}{\lambda^5} \left( \frac{1-\phi}{1-\phi_0} \right)^{2/3} \frac{\phi}{\phi_0}. \tag{1}$$

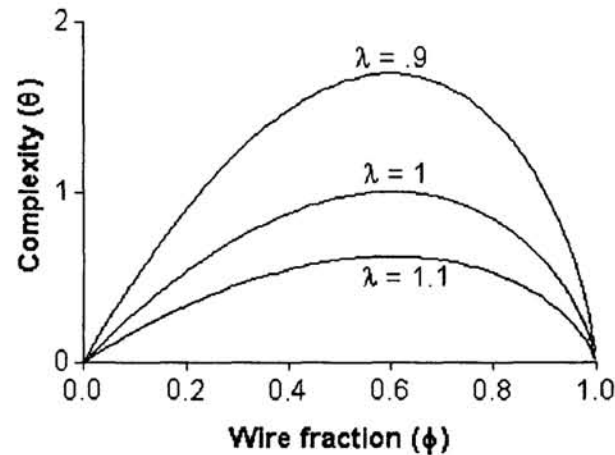

Figure 1: Relative complexity ($\theta$) as a function of volume wire fraction ($\phi$). The graphs are calculated from equation (1) for three values of the parameter $\lambda$ as indicated; this parameter determines the average length of wire associated with a synapse (relative to this length for the real cortex, for which ($\lambda = 1$). Note that as the average length of wire per synapse increases, the maximum possible complexity decreases.

For the following discussion assume that $\lambda = 1$; we return to the meaning of this parameter later. To derive this equation two assumptions are made. First, we suppose that each added synapse requires extra wire equal to the average wire length and volume per synapse in the unperturbed cortex. Second, because adding wire for new synapses increases the brain volume and therefore increases the distance axons and dendrites must travel to maintain the connections they make in the real cortex, all of the dendrite and unmyelinated axon diameters are increased in proportion to the square of their length changes in order to maintain the intersynaptic conduction times[2] and dendrite cable lengths[3] as they are in the actual cortex. If the unmyelinated axon diameters were not increased as the axons become longer, for example, the time for a nerve impulse to propagate from one synapse to the next would be increased and we would violate our rule that the existing circuit and its function be unchanged. We note that the vast majority of cortical axons are unmyelinated.[1] The plot of $\theta$ as a function of $\phi$ is parabolic-like (see Figure 1) with a maximum value at $\phi = 0.6$, a point at which $d\theta/d\phi = 0$. This same maximum value is found for any possible value of $\phi_0$, the real cortical wire fraction.

Why does complexity reach a maximum value at a particular wire fraction? When wire and synapses are added, a series of consequences can lead to a runaway situation we call the *wiring catastrophe*. If we start with a wire fraction less than 0.6, adding wire increases the cortical volume, increased volume makes longer paths for axons to reach their targets which requires larger diameter wires (to keep conduction delays or cable attenuation constant from one point to another), the larger wire diameters increase cortex volume which means wires must be longer, etc. While the wire fraction $\phi$ is less than 0.6, increasing complexity is accompanied by finite increases in $\phi$. At $\phi = 0.6$ the rate at which wire fraction increases with complexity becomes infinite $d\phi/d\theta \to \infty$); we have reached the wiring catastrophe. At this point, adding wire becomes impossible without decreasing complexity or making other changes – like decreasing axon diameters – that alter cortical function. The physical cause of the catastrophe is a slow growth of conduction velocity and dendritic cable length with diameter combined with the requirement that the conduction times between synapses (and dendrite cable lengths) be unchanged in the perturbed cortex.

We assumed above that each synapse requires a certain amount of wire, but what if we could

add new synapses using the wire already present? We do not know what factors determine the wire volume needed to support a synapse, but if the average amount of wire per synapse could be less (or more) than that in the actual cortex, the maximum wire fraction would still be 0.6. Each curve in Figure 1 corresponds to a different assumed average wire length required for a synapse (determined by $\lambda$), and the maximum always occurs at 0.6 independent of $\lambda$. In the following we consider only situations in which $\lambda$ is fixed.

For a given $\lambda$, what complexity should we expect for the actual cortex? Three arguments favor the maximum possible complexity. The greatest complexity gives the largest number of synapses per neuron and this permits more bits of information to be represented per neuron. Also, more synapses per neuron decreases the relative effect caused by the loss or malfunction of a single synapse. Finally, errors in the local wire fraction would minimally affect the local complexity because $d\theta/d\phi = 0$ at $\phi = 0.6$. Thus one can understand why the actual cortex has the wire fraction we identify as optimal.[1]

This conclusion that the wire fraction is a maximum in the real cortex has an interesting consequence: components of an actual cortical circuit cannot be rearranged in a way that needs more wire without eliminating synapses or reducing wire diameters. For example, if intermixing the cell bodies of left and right eye cells in primate primary visual cortex (rather than separating them in ocular dominance columns) increased the average length of the wire[4] the existing circuit could not be maintained just by a finite increase in volume. This happens because a greater wire length demanded by the rearrangement of the same circuit would require longer wire per synapse, that is, an increased $\lambda$. As can be seen from Figure 1, brains with $\lambda > 1$ can never achieve the complexity reached at the maximum of the $\lambda = 1$ curve that corresponds to the actual cortex.

Our observations support the notion that brains are arranged to minimize wire length. This idea, dating back to Cajal[5], has recently been used to explain why retinotopic maps exist[6],[7], why cortical regions are separated, why ocular dominance columns are present in primary visual cortex[4],[8],[9] and why the cortical areas and flat worm ganglia are placed as they are.[10-13] We anticipate that maximal complexity/minimal wire length arguments will find further application in relating functional and anatomical properties of brain.

## Methods

The volume of the cube of cortex we perturb is $V$, the volume of the non-wire portion is $W$ (assumed to be constant), the fraction of $V$ consisting of wires is $\phi$, the total number of synapses is $N$, the average length of axonal wire associated with each synapse is $s$, and the average axonal wire volume per unit length is $h$; the corresponding values for dendrites are indicated by primes ($s'$ and $h'$). The unperturbed value for each variable has a 0 subscript; thus the volume of the cortical cube before it is perturbed is

$$V_0 = W_0 + N_0(s_0 h_0 + s_0' h_0').$$

(2)

We now define a "virtual" perturbation that we use to explore the extent to which the actual cortical region contains an optimal fraction of wire. If we increase the number of synapses by a factor $\theta$ and the length of wire associated with each synapse by a factor $\lambda$, then the perturbed cortical cube's volume becomes

$$V_0 = W_0 + \lambda\theta \left( N_0 s_0 h_0 \frac{h}{h_0} + N_0 s_0' h_0' \frac{h'}{h_0'} \right) (V/V_0)^{1/3}.$$

(3)

This equation allows for the possibility that the average wire diameter has been perturbed and increases the length of all wire segments by the "mean field" quantity $(V/V_0)^{1/3}$ to take account of the expansion of the cube by the added wire; we require our perturbation disperses the added wire as uniformly as possible throughout the cortical cube.

To simplify this relation we must eliminate $h/h_0$ and $h'/h'_0$; we consider these terms in turn. When we perturb the brain we require that the average conduction time ($s/u$, where $u$ is the conduction velocity) from one synapse to the next be unchanged so that $s/u = s_0/u_0$, or

$$\frac{u}{u_0} = \frac{s}{s_0} = \frac{\lambda s_0 (V/V_0)^{1/3}}{s_0} = \lambda (V/V_0)^{1/3}. \tag{4}$$

Because axon diameter is proportional to the square of conduction velocity $u$ and the axon volume per unit length $h$ is proportional to diameter squared, $h$ is proportional to $u^4$ and the ratio $h/h_0$ can be written as

$$\frac{h}{h_0} = \left(\frac{u}{u_0}\right)^4 = \frac{s}{s_0} = \lambda^4 (V/V_0)^{4/3}. \tag{5}$$

For dendrites, we require that their length from one synapse to the next in units of the cable length constant be unchanged by the perturbation. The dendritic length constant is proportional to the square root of the dendritic diameter $d$, so $s/\sqrt{d} = s_0/\sqrt{d_0}$ or

$$\frac{d}{d_0} = \left(\frac{s'}{s'_0}\right)^2 = \left(\lambda (V/V_0)^{1/3}\right)^2 = \lambda^2 (V/V_0)^{2/3}. \tag{6}$$

Because dendritic volumes per unit length ($h$ and $h'$) vary as the square of the diameters, we have that

$$\frac{h'}{h'_0} = \left(\frac{d}{d_0}\right)^2 = \lambda^4 (V/V_0)^{4/3}. \tag{7}$$

The equation (2) can thus be rewritten as

$$V = W_0 + N_0(s_0 h_0 + s'_0 h'_0)\theta\lambda^5 (V/V_0)^{5/3}. \tag{8}$$

Divide this equation by $V_0$, define $v = V/V_0$, and recognize that $W_0/V_0 = (1 - \phi_0)$ and that $\phi_0 = N_0(s_0 h_0 + s'_0 h'_0)/V_0$ ; the result is

$$\nu = (1 - \phi_0) + \theta\lambda^5 \phi_0 \nu^{5/3}. \tag{9}$$

Because the non-wire volume is required not to change with the perturbation, we know that $W_0 = (1 - \phi_0)V_0 = (1 - \phi)V$ which means that $\nu = (1 - \phi_0)/(1 - \phi)$; substitute this in equation (9) and rearrange to give

$$\theta = \frac{1}{\lambda^5} \left(\frac{1-\phi}{1-\phi_0}\right)^{2/3} \frac{\phi}{\phi_0}. \tag{1}$$

the equation used in the main text.

We have assumed that conduction velocity and the dendritic cable length constant vary exactly with the square root of diameter[2],[14] but if the actual power were to deviate slightly from 1/2 the wire fraction that gives the maximum complexity would also differ slightly from 0.6.

## Acknowledgments

This work was supported by the Howard Hughes Medical Institute and a grant from NIH to C.F.S. D.C. was supported by a Sloan Fellowship in Theoretical Neurobiology.

# References

[1] Braitenberg, V. & Schuz, A. *Cortex: Statistics and Geometry of Neuronal Connectivity* (Springer, 1998).

[2] Rushton, W.A.H. A Theory of the Effects of Fibre Size in Medullated Nerve. J. Physiol. 115, 101-122 (1951).

[3] Bekkers, J.M. & Stevens, C.F. Two different ways evolution makes neurons larger. Prog Brain Res 83, 37-45 (1990).

[4] Mitchison, G. Neuronal branching patterns and the economy of cortical wiring. Proc R Soc Lond B Biol Sci 245, 151-158 (1991).

[5] Cajal, S.R.Y. *Histology of the Nervous System* 1-805 (Oxford University Press, 1995).

[6] Cowey, A. Cortical maps and visual perception: the Grindley Memorial Lecture. Q J Exp Phychol 31, 1-17 (1979).

[7] Allman J.M. & Kaas J.H. The organization of the second visual area (V II) in the owl monkey: a second order transformation of the visual hemifield. Brain Res 76: 247-65 (1974).

[8] Durbin, R. & Mitchison, G. A dimension reduction framework for understanding cortical maps. Nature 343, 644-647 (1990).

[9] Mitchison, G. Axonal trees and cortical architecture. Trends Neurosci 15, 122-126 (1992).

[10] Young, M.P. Objective analysis of the topological organization of the primate cortical visual system. Nature 358, 152-154 (1992).

[11] Cherniak, C. Local optimization of neuron arbors. Biol Cybern 66, 503-510 (1992).

[12] Cherniak, C. Component placement optimization in the brain. J Neurosci 14, 2418-2427 (1994).

[13] Cherniak, C. Neural component placement. Trends Neurosci 18, 522-527 (1995).

[14] Rall, W. in *Handbook of Physiology, The Nervous Systems, Cellular Biology of Neurons* (ed. Brookhart , J.M.M., V.B.) 39-97 (Am. Physiol. Soc., Bethesda, MD, 1977).
